# Inferring rankings under constrained sensing

**Srikanth Jagabathula**    **Devavrat Shah**

Laboratory of Information and Decision Systems,
Massachusetts Institute of Technology,
Cambridge, MA 02139.
{jskanth, devavrat}@mit.edu

## Abstract

Motivated by applications like elections, web-page ranking, revenue maximization etc., we consider the question of inferring *popular* rankings using *constrained* data. More specifically, we consider the problem of inferring a probability distribution over the group of permutations using its first order marginals. We first prove that it is not possible to recover more than $O(n)$ permutations over $n$ elements with the given information. We then provide a simple and novel algorithm that can recover up to $O(n)$ permutations under a natural stochastic model; in this sense, the algorithm is optimal. In certain applications, the interest is in recovering only the most popular (or mode) ranking. As a second result, we provide an algorithm based on the Fourier Transform over the symmetric group to recover the mode under a natural majority condition; the algorithm turns out to be a maximum weight matching on an appropriately defined weighted bipartite graph. The questions considered are also thematically related to Fourier Transforms over the symmetric group and the currently popular topic of *compressed sensing*.

## 1 Introduction

We consider the question of determining a real-valued function on the space of permutations of $n$ elements with very limited observations. Such a question naturally arises in many applications including efficient web-page rank aggregation, choosing the winner in a sport season, setting odds in gambling for revenue maximization, estimating popularity of candidates pre-election and the list goes on (for example, see references [1], [2], [3]). In what follows, we give a motivating example for the pursuit of this quest.

**A motivating example.** Consider a pre-election scenario in a democratic country with $n$ potential candidates. Each person (or voter) has certain ranking of these candidates in mind (consciously or sub-consciously). For example, let $n = 3$ and the candidates be $A, B$ and $C$. Each person believes in one of the $3! = 6$ possible ordering of these candidates. For example, let $50\%$ of people believe in $A > B > C$, $30\%$ of people believe in $B > A > C$ and $20\%$ of people believe in $C > A > B$. We wish to infer these preferences of population by means of a limited set of questions.

Specifically, suppose we can interview a representative collection (i.e. reasonably large random collection) of people for this purpose. However, in the interview we may not be able to ask them their complete ranking of all candidates. This may be because a person may not be able to articulate it clearly. Or, in situations (e.g. gambling) where there is a financial significance associated with information of complete ranking, an individual may not be ready to provide that information. In such a situation, we will have to settle with restricted questions of the following type: *what will be the rank of candidate A in your opinion*? or, *whom would you rank second*?

Given answers to such restricted questions, we would like to infer what fraction of the population prefers which ordering of candidates. Clearly, such restricted information cannot lead to any useful

inference of prevalent ordering of candidates in the population if there are too many of them (for large $n$). Now, in a real world scenario, it is likely that people decide rankings of candidates based on a few issues such as war, abortion, economy and gay marriage. That is, an individual will decide the ranking of the candidates based on the opinions of candidates on these issues. Therefore, irrespective of the number of candidates, the number of distinct rankings that prevail in the population are likely to be very few.

In this paper, we are interested in inferring such *few* prevalent rankings of candidates and their popularity based on the restricted (or partial) information as explained above. Thematically, this question is similar to the pursuit of *compressed sensing*. However, as we explain in Section 2, standard compressed sensing does not apply under this setting. We also discuss a natural relation between the available information and the Fourier coefficients of the Fourier transformation based on group representation (see Proposition 1). It turns out that the problem we consider is equivalent to that of recovery of a function over a symmetric group using the *first order* Fourier coefficients. Thus, our problem is thematically related to the recovery of functions over non-commutative groups using a limited set of Fourier coefficients. As we show in Section 2, a naive recovery by setting the unknown Fourier coefficients to zero yields a very bad result. Hence, our approach has potential applications to yielding a better recovery.

In many applications, one is specifically interested in finding out the most popular ranking (or mode) rather than all the prevalent rankings. For this, we consider an approximation based on Fourier transformation as a surrogate to find the mode. We establish that under the natural *majority* condition, our algorithm finds the correct mode (see Theorem 2). Interestingly enough, our algorithm to find an estimate of the mode corresponds to finding a maximum weight matching in a weighted bipartite graph of $n$ nodes.

**Organization.** We start describing the setup, the problem statement, and the relation to compressed sensing and Fourier transform based approaches in Section 2. In Section 3, we provide precise statements of the main results. In the remaining Sections, we prove these results and discuss the relevant algorithms.

## 2   Background and preliminaries

**Setup.** Let $S_n = \{\sigma_1, \ldots, \sigma_N\}$ denote set of all possible $N = n!$ permutations (orderings) of $n$ elements. $S_n$ is also known as the *symmetric group* of degree $n$. Let $f \colon S_n \to [0,1]$ denote a mapping from the symmetric group to the interval $[0,1]$. We assume that the function $f$ is normalized i.e., $\|f\|_{\ell_1} = 1$, where $\|\cdot\|_{\ell_1}$ denotes the $\ell_1$ norm. Let $p_k$ denote the value $f(\sigma_k)$, for $1 \leq k \leq N$. Without loss of generality we assume that the permutations are labeled such that $p_k \leq p_m$ for $k < m$. We write $f(\cdot)$ to denote the function and $f$ to denote the vector $(f(\sigma_k))_{N \times 1}$. The set of permutations for which $f(\cdot)$ is non-zero will be called the *support* of $f(\cdot)$; also, the cardinality of the support will be called *sparsity* of $f$ and is denoted by $K$ i.e., $K = \|f\|_{\ell_0}$. Each permutation $\sigma$ will be represented by its corresponding permutation matrix denoted by $P^\sigma$ i.e., $P^\sigma_{ij} = \mathbb{1}_{\{\sigma(j)=i\}}$, where $\mathbb{1}_E$ is the indicator variable of the event $E$. For brevity, we write $P^\sigma$ to mean both the $n \times n$ matrix and the $n^2 \times 1$ vector. We use the terms permutation and permutation matrix interchangeably. We think of permutations as complete matchings in a bipartite graph. Specifically, we consider an $n \times n$ bipartite graph and each permutation corresponds to a complete matching in the graph. The edges in a permutation will refer to the edges in the corresponding bipartite matching. For $1 \leq i, j \leq n$, let

$$q_{ij} := \sum_{\sigma \in S_n : \sigma(j) = i} f(\sigma) \tag{1}$$

Let $Q$ denote both the matrix $(q_{ij})_{n \times n}$ and the vector $(q_{ij})_{n^2 \times 1}$. It is easy to note that $Q$ can be equivalently written as $\sum_{\sigma \in S_n} f(\sigma) P^\sigma$. From the definition, it also follows that $Q$ is a doubly stochastic matrix. The matrix $Q$ corresponds to the *first order* information about the function $f(\cdot)$. In the election example, it is easy to see that $q_{ij}$ denotes the fraction of voters that have ranked candidate $j$ in the $i^{\text{th}}$ position.

**Problem statement and result.** The basic objective is to determine the values of the function $f(\cdot)$ precisely, using only the values of the matrix $Q$. We will first prove, using information theoretic techniques, that recovery is *asymptotically reliable* (average probability of error goes to zero as

$n \to \infty$) only if $K = O(n)$. We then provide a novel algorithm that recovers prevalent rankings and their popularity exactly under minimal (essentially necessary) conditions; under a natural stochastic model, this algorithm recovers up to $O(n)$ permutations. In this sense, our algorithm is optimal.

It is often the case that the full knowledge of functional values at all permutations is not required. Specifically, in scenarios such as ranked elections, interest is in finding the most likely permutation i.e., $\arg \max f(\sigma)$. Theorem 2 proves that the max-weight matching yields the most likely permutation under natural *majority* assumption.

## 2.1 Relation to Fourier Transform

The question we consider is thematically related to harmonic analysis of functions over non-commutative groups. As we shall show soon, the matrix $Q$ is related to the first two Fourier coefficients of the Fourier Transform of the distribution over the permutation group. Thus, the problem we are considering can be restated as that of reconstructing a distribution over the permutation group from its first two Fourier coefficients. Reconstructing distributions over the permutation group from a limited number of Fourier coefficients has several applications. Specifically, there has been some recent work on *multi-object tracking* (see [4] and [3]), in which they approach the daunting task of maintaining a distribution over the permutation group by approximating it using the first few Fourier coefficients. This requires reconstructing the function from a limited number of Fourier coefficients, where our solution can be potentially applied.

We will now discuss the Fourier Transform of a function on the permutation group, which provides another possible approach for recovery of $f$. Interestingly enough, the *first order* Fourier transform of $f$ can be constructed using information based on $Q = (q_{ij})$. As we shall find, this approach fails to recover sparse $f$ as it has tendency to "spread" the mass on all $n!$ elements given $Q$. However, as established in Theorem 2 this leads to recovery of *mode* or most likely assignment of $f$ under natural *majority* condition.

Next, some details on what the Fourier transform (an interested reader is requested to check [5] for missing details) based approach is, how $Q$ can be used to obtain an approximation of $f$ and why it does not recover $f$ exactly. The details relevant to recovery of mode of $f$ will be associated with Theorem 2.

**Fourier Transform: Definition.** We can obtain *a* solution to the set of linear equations in (8) using the Fourier Transforms at symmetric group representations. For a function $h \colon G \to \mathbb{R}$ on group $G$, its Fourier Transform at a representation $\rho$ of $G$ is defined as $\hat{h}_\rho = \sum_\sigma h(\sigma)\rho(\sigma)$. The collection of Fourier Transforms of $h(\cdot)$ at a complete set of inequivalent irreducible representations of $G$ completely determine the function. This follows from the following expression for the *inverse* Fourier Transform:

$$h(\sigma) = \frac{1}{|G|} \sum_k d_{\rho_k} \operatorname{Tr}\left[ \hat{h}_{\rho_k}^T \rho_k(\sigma) \right] \tag{2}$$

where $|G|$ denotes the cardinality of $G$, $d_{\rho_k}$ denotes the degree of representation $\rho_k$ and $k$ indexes over the complete set of inequivalent irreducible representations of $G$. The trivial representation of a group is the 1-dimensional representation $\rho_0(\sigma) = 1, \forall \sigma \in G$. Therefore, the Fourier Transform of $h(\cdot)$ at $\rho_0$ is $\sum_\sigma h(\sigma)$.

**Fourier Transform: Approximation.** The above naturally suggests an *approximation* based on a limited number of *Fourier coefficients* with respect to a certain subset of irreducible representations. We will show that, indeed, the information matrix $Q$ corresponds to the Fourier coefficient with respect to the *first-order* representation of the symmetric group $S_n$. Therefore, it yields a natural approximation.

It is known that [5] the *first order permutation representation* of $S_n$, denoted by $\tau_1$, has a degree $n$ and maps every permutation $\sigma$ to its corresponding permutation matrix $P^\sigma$. In other words, we have $\tau_1(\sigma) = P^\sigma$. Thus, $\hat{f}(\sigma) = \sum_{\sigma \in S_n} f(\sigma)\tau_1(\sigma) = Q$. Reconstruction of $f$ requires Fourier Transforms at irreducible representations. Even though $\tau_1$ is not an irreducible representation, it is known that [5] that every representation of a group is equivalent to the direct sum of irreducible representations. In particular, $\tau_1$ can be decomposed into $\tau_1 = \rho_0 \oplus \rho_1$; where $\rho_0$ is the aforementioned trivial representation of degree 1 and $\rho_1$ is an irreducible representation of degree $n - 1$. It is worth pointing out to a familiar reader that what we call $\rho_1$ is more appropriately denoted by $\rho_{(n-1,1)}$ in

the literature; but we will stick to $\rho_1$ for brevity. Thus, $Q$ is related to the Fourier Transforms of the irreducible representations $\rho_0$ and $\rho_1$. We now have the following proposition:

**Proposition 1.** *Consider a function* $f : S_n \rightarrow \mathbb{R}$*. Suppose that* $\|f\|_{\ell_1} = 1$ *and we are given its corresponding* $Q$*. Then, its natural Fourier approximation obtained by looking at the Fourier coefficients of the relevant irreducible representations is given by the function* $\tilde{f} : S_n \rightarrow \mathbb{R}$ *defined as:*

$$\tilde{f}(\sigma) = (n-1)\frac{\langle Q, P^\sigma \rangle}{N} - \frac{n-2}{N} \tag{3}$$

*for* $\sigma \in S_n$*, with* $N = n!$*,* $\|f\|_{\ell_1} = \|\tilde{f}\|_{\ell_1}$ *and* $\sum_{\sigma \in S_n} \tilde{f}(\sigma)P^\sigma = Q$*.*

*Proof.* We have:

$$Q = \sum_{\sigma \in S_n} f(\sigma)\tau_1 = \sum_{\sigma \in S_n} f(\sigma)(\tau_0 \oplus \tau_1) = \hat{f}_{\rho_0} \oplus \hat{f}_{\rho_1}. \tag{4}$$

Therefore,

$$\langle Q, P^\sigma \rangle = \mathrm{Tr}\left[Q^T P^\sigma\right] = \mathrm{Tr}\left[\left(\hat{f}_{\rho_0}^T \oplus \hat{f}_{\rho_1}^T\right)(\rho_0(\sigma) \oplus \rho_1(\sigma))\right] \tag{5}$$

Since Tr is independent of the basis, choosing an appropriate basis we can write:

$$\langle Q, P^\sigma \rangle = \mathrm{Tr}\left[\hat{f}_{\rho_0}^T \rho_0(\sigma)\right] + \mathrm{Tr}\left[\hat{f}_{\rho_1}^T \rho_1(\sigma)\right] = 1 + \mathrm{Tr}\left[\hat{f}_{\rho_1}^T \rho_1(\sigma)\right] \tag{6}$$

(6) is true because $\rho_0(\sigma) = 1, \forall \sigma \in S_n$, and $\|f\|_{\ell_1} = 1$.

$\tilde{f}$ is obtained by truncating the Inverse Fourier Transform expression to the first two terms. Thus, from (2), it follows that:

$$\tilde{f}(\sigma) = \frac{1}{N}\left[\hat{f}_{\rho_o}^T \rho_0(\sigma) + (n-1)\hat{f}_{\rho_1}^T \rho_1(\sigma)\right] \tag{7}$$

Using the fact that $\rho_0(\sigma) = 1 \; \forall \sigma \in S_n$, $\hat{f}_{\rho_0} = 1$, and plugging (6) into (7) gives the result of the proposition. $\qquad \square$

**Summary.** Thus, the Fourier Transform technique yields *a* solution to the problem. Unfortunately, the solution is not sparse and the "mass" is distributed over all the permutations yielding values of $O(1/N)$ for all permutations. In summary, a naive approach to the reconstruction of a sparse distribution gives unsatisfactory results and requires a different approach.

## 2.2 Relation to Compressed Sensing

Here we discuss the relation of the above stated question to the recently popular topic of *compressed sensing*. Indeed, both share the commonality in the sense that the ultimate goal is to recover a sparse function (or vector) based on few samples. However, as we shall show, the setup of our work here is quite different. This is primarily because in the standard compressed sensing setup, samples are chosen as "random projections" while here samples are highly constrained and provide information matrix $Q$. Next, we provide details of this.

Our problem can be formulated as a solution to a set of linear equations by defining a matrix $A$ as the $n^2 \times N$ matrix with column vectors as $P^{\sigma_k}, 1 \leq k \leq N$. Then, $f$ is a solution to the following set of linear equations:

$$Ax = Q \tag{8}$$

Candes and Tao (2005) [6] provide an approach to solve this problem. They require the vector $f$ to be sparse i.e., $\|f\|_{\ell_0} = \rho N$, for some $\rho > 0$. As discussed earlier, this is a reasonable assumption in our case because: (a) the total number of permutations $N$ can be very large even for a reasonably sized $n$ and (b) most functions $f(\cdot)$ that arise in practice are determined by a small (when compared to $N$) number of parameters. Under a restriction on the isometry constants of the matrix $A$, Candes and Tao prove that the solution $f$ is the unique minimizer to the LP:

$$\min \|x\|_{\ell_1} \quad \text{s.t.} \quad Ax = Q \tag{9}$$

Unfortunately, the approach of Candes and Tao cannot be directly applied to our problem because the isometry constants of the matrix $A$ do not satisfy the required conditions.

We now take a closer look at the isometry constants of $A$. Gaussian random matrices form an important class of matrices with good isometry constants. Unfortunately, neither is our matrix $A$ random nor is there a straightforward random formulation of our problem. To see why the matrix $A$ has bad isometry constants, we take a simple example. For any $n \geq 4$ consider the following 4 permutations: $\sigma_1 = \text{id}$, $\sigma_2 = (12)$, $\sigma_3 = (34)$ and $\sigma_4 = (12)(34)$. Here, id refers to the identity permutation and the permutations are represented using the cycle notation. It is easy to see that:

$$P^{\sigma_1} + P^{\sigma_4} = P^{\sigma_2} + P^{\sigma_3} \tag{10}$$

For any integer $1 \leq S \leq N$, the $S$ restricted isometry constant of $A$ is defined as the smallest quantity such that $A_T c$ obeys:

$$(1 - \delta_S)\|c\|_{\ell_2}^2 \leq \|A_T c\|_{\ell_2}^2 \leq (1 + \delta_S)\|c\|_{\ell_2}^2 \tag{11}$$

$\forall\ T \subseteq \{1, 2, \ldots, N\}$ of cardinality at most $S$ and all real vectors $c$. Here, $A_T c$ denotes $\sum_{k \in T} c_k P^{\sigma_k}$. From this definition and (10), it follows that $\delta_S = 1\ \forall\ S \geq 4$. Theorem 1.4 requires $\delta_S < 1$ for perfect reconstruction of $f$ when $\|f\|_{\ell_0} \leq S$. Therefore, the compressed sensing approach of Candes and Tao does not guarantee the unique reconstruction of $f$ if $\|f\|_{\ell_0} \geq 4$.

## 3 Main results

**Exact recovery.** The main result of this paper is about the exact recovery of $f$ from the given constrained information matrix $Q = (q_{ij})$ under the hypothesis that $f$ is sparse or has small $\|f\|_{\ell_0}$. We provide an algorithm that recovers $f$ exactly if the underlying support and probabilities have the following two properties:

**Property 1** (P1). *Suppose the function $f(\cdot)$ is $K$ sparse. Let $p_1, p_2, \ldots, p_K$ be the function values. The following is true:*

$$\sum_{j \in J} p_j \neq \sum_{j \in J'} p_j\ \forall\ J, J' \subseteq \{1, 2, \ldots, K\}\ s.t\ J \cap J' = \emptyset$$

**Property 2** (P2). *Let $\{\sigma_1, \sigma_2, \ldots, \sigma_K\}$ be the support of $f(\cdot)$. For each $1 \leq i \leq K$, $\exists$ an $1 \leq \eta_i \leq n$ such that $\sigma_i(\eta_i) \neq \sigma_j(\eta_i)\ \forall\ j \neq i$. In other words, each permutation has at least one edge that is different from all the others.*

When properties P1 and P2 are satisfied, the equation $Q = Af$ has a unique solution and can indeed be recovered; we will provide an algorithm for such recovery. The following is the formal statement of this result and will be proved later.

**Theorem 1.** *Consider a function $f \colon S_n \to [0, 1]$ such that $\|f\|_{\ell_0} = L$, $\|f\|_{\ell_1} = 1$, and the functional values and the support possess properties P1 and P2. Then, matrix $Q$ is sufficient to reconstruct $f(\cdot)$ precisely.*

**Random model, Sparsity and Theorem 1.**

Theorem 1 asserts that when properties P1 and P2 are satisfied, exact recovery is possible. However, it is not clear why they are reasonable. We will now provide some motivation and prove that the algorithm is indeed optimal in terms of the maximum sparsity it can recover.

Let's go back to the counter-example we mentioned before: For any $n \geq 4$ consider the 4 permutations $\sigma_1 = \text{id}, \sigma_2 = (12), \sigma_3 = (34)$ and $\sigma_4 = (12)(34)$. We have $P^{\sigma_1} + P^{\sigma_4} = P^{\sigma_2} + P^{\sigma_3}$. Now, consider 4 values $p_1, p_2, p_3$ and $p_4$. Without loss of generality suppose that $p_1 \leq p_4$ and $p_2 \leq p_3$. Using the equation $P^{\sigma_1} + P^{\sigma_4} = P^{\sigma_2} + P^{\sigma_3}$, we can write the following:

$$\begin{aligned}
Q &= p_1 P^{\sigma_1} + p_2 P^{\sigma_2} + p_3 P^{\sigma_3} + p_4 P^{\sigma_4} \\
&= (p_1 + p_2) P^{\sigma_1} + (p_1 + p_2) P^{\sigma_4} + (p_3 - p_2) P^{\sigma_3} \\
&= (p_1 + p_2) P^{\sigma_2} + (p_1 + p_3) P^{\sigma_3} + (p_4 - p_1) P^{\sigma_4}.
\end{aligned}$$

Thus, under the above setup, there is no unique solution to $Q = Af$. In addition, from the last two equalities, we can conclude that even the *sparsest* solution is not unique. Hence, there is no hope of recovering $f$ given only $Q$ in this setup.

The question we now ask is whether the above counter example is contrived and specially constructed, or is it more prevalent. For that, we consider a random model which puts a uniform measure on all the permutations. The hope is that under this model, situations like the counter example occur with a vanishing probability. We will now describe the random model and then state important results on the sparsity of $f$ that can be recovered from $Q$.

*Random Model.* Under the random model, we assume that the function $f$ with sparsity $K$ is constructed as follows: Choose $K$ permutations uniformly at random and let them have any non-trivial real functional values chosen uniformly at random from a bounded interval and then normalized.

We call an algorithm producing an estimate $\hat{f}$ of $f$ as asymptotically reliable if $\Pr\left[f \neq \hat{f}\right] = \varepsilon(n)$ where $\varepsilon(n) \to 0$ as $n \to \infty$. We now have the following two important results:

**Lemma 1.** *Consider a function $f \colon S_n \to \mathbb{R}$ with sparsity $K$. Given the matrix $Q = Af$, and no additional information, the recovery will be asymptotically reliable only if $K \leq 4n$.*

First note that a trivial bound of $(n-1)^2$ can be readily obtained as follows: Since $Q$ is doubly stochastic, it can be written as a convex combination of permutation matrices [7], which form a space of dimension $(n-1)^2$. Lemma 1 says that this bound is loose. It can be proved using standard arguments in Information Theory by considering $A$ as a channel with input $f$ and output $Q$.

**Lemma 2.** *Consider a function $f \colon S_n \to \mathbb{R}$ with sparsity $K$ constructed according to the random model described above. Then, the support and functional values of $f$ possess properties P1 and P2 with probability $1 - o(1)$ as long as $K \leq 0.6n$.*

It follows from Lemma 2 and Theorem 1 that $f$ can be recovered exactly from $Q$ if the sparsity $K = O(n)$. Coupled with Lemma 1 we conclude that our algorithm is optimal in the sense that it achieves the sparsity bound of $O(n)$.

**Recovery of Mode.** As mentioned before, often we are interested in obtaining only limited information about $f(\cdot)$. One such scenario is when we would like to find just the most likely permutation. For this purpose, we use the Fourier approximation $\tilde{f}$ (cf. Proposition 1) in place of $f$: that is, the mode of $f$ is estimated as mode of $\tilde{f}$. The following result states the correctness of this approximation under *majority*.

**Theorem 2.** *Consider a function $f \colon S_n \to [0,1]$ such that $\|f\|_{\ell_0} = L$ and $\|f\|_{\ell_1} = 1$. Suppose the majority condition holds, that is $\max_{\sigma \in S_n} f(\sigma) > 1/2$. Then,*

$$\arg\max_{\sigma \in S_n} f(\sigma) = \arg\max_{\sigma \in S_n} \tilde{f}(\sigma) = \arg\max_{\sigma \in S_n} \langle P^\sigma, Q \rangle .$$

The mode of $\tilde{f}$, or maximizer of $\langle P^\sigma, Q \rangle$ is essentially the maximum weight matching in a weighted bipartite graph: consider a complete bipartite graph $G = ((V_1, V_2), E)$ with $V_1 = V_2 = \{1, \ldots, n\}$ and $E = V_1 \times V_2$ with edge $(i,j) \in E$ having weight $q_{ij}$. Then, weight of a matching (equivalently permutation $\sigma$) is indeed $\langle P^\sigma, Q \rangle$. The problem of finding maximum weight matching is classical. It can be solved in $O(n^3)$ using algorithm due to Edmond and Karp [8] or max-product belief propagation by Bayati, Shah and Sharma [9]. Thus, this is an approximation that can be evaluated.

## 4 Theorem 1: Proof and Algorithm

Here, we present a constructive proof of Theorem 1. Specifically, we will describe an algorithm to determine the function values from $Q$ which will be the original $f$ as long as properties P1 and P2 are satisfied.

Let $p_1, p_2, \ldots, p_L$ denote the non-zero functional values. Let $\sigma_1, \sigma_2, \ldots, \sigma_L$ denote the corresponding permutations i.e., $f(\sigma_k) = p_k$. Without loss of generality assume that the permutations are labeled such that $p_i \leq p_j$ for $i < j$. Let $q_1, q_2, \ldots, q_M$, where $M = n^2$, denote the values of matrix $Q$ arranged in ascending order.

Given this sorted version, we have $q_i \leq q_j$ for $i < j$. Let $e_i$ denote the edge $(u, v)$ such that $q_i = q_{e_i} = q_{uv}$, where recall that

$$q_{uv} = \sum_{k:\sigma_k(u)=v} f(\sigma_k) = \sum_{k:\sigma_k(u)=v} p_k.$$

Let $A_k$ denote the set of edges corresponding to permutation $\sigma_k$, $1 \leq k \leq L$. That is, $A_k = \{(u, \sigma_k(u)) : 1 \leq u \leq n\}$. The algorithm stated below will itself determine $L$, and $(A_k, p_k)1 \leq k \leq L$ using information $Q$. The algorithm works when properties P1 and P2 are satisfied.

---

**Algorithm:**

---

*initialization*: $p_0 = 0, k(0) = 0$ and $A_k = \emptyset, 1 \leq k \leq M$.
*for*   $i = 1$ *to*  $M$
    *if*   $q_i = \sum_{j \in J} p_j$  *for  some*  $J \subseteq \{0, 1, \ldots, k(i-1)\}$
        $k(i) = k(i-1)$
        $A_j = A_j \cup \{e_i\}$  $\forall$  $j \in J$
    *else*
        $k(i) = k(i-1) + 1$
        $p_{k(i)} = q_i$
        $A_{k(i)} = A_{k(i)} \cup \{e_i\}$
    *end if*
*end for*
*Output* $L = k(i)$ and $(p_k, A_k), 1 \leq k \leq L$.

---

By property P2, there exists at least one $q_i$ such that it is equal to $p_k$, for each $1 \leq k \leq L$. The property P1 ensures that whenever $q_i = p_{k(i)}$, the condition in the "if" statement of the pseudocode is not satisfied. Therefore, the algorithm correctly assigns values to each of the $p_k$'s.

Note that the condition in the "if" statement being true implies that edge $e_i$ is present in all the permutations $\sigma_j$ such that $j \in J$. Property P1 ensures that such a $J$, if exists, is unique. Therefore, when the condition is satisfied, the only permutations that contain edge $e_i$ are $\sigma_j, j \in J$.

When the condition in the "if" statement fails, again from properties P1 and P2 it follows that edge $e_i$ is contained only in permutation $\sigma_{k(i)}$. From this discussion we can conclude that at the end of the iterations, each of the $A_i$'s contain complete information about their corresponding permutations.

The algorithm thus completely determines the function $f(\cdot)$. Finally, note that the algorithm does not require the knowledge of $\|f\|_{\ell_0}$.

## 5  Theorem 2: Proof and Algorithm

Here, our interest is in finding the mode of $f$. The algorithm we have proposed is use the mode of $\tilde{f}$, as an estimate of mode of $f$. We wish to establish that when $\max_{\sigma \in S_n} f(\sigma) > 1/2$ then

$$\tilde{\sigma}^* = \sigma^*, \quad \text{where} \quad \tilde{\sigma}^* = \arg\max_{\sigma \in S_n} \tilde{f}(\sigma); \qquad \sigma^* = \arg\max_{\sigma \in S_n} f(\sigma).$$

Since we have assumed that $f(\sigma^*) > 1/2$ and $\|f\|_{\ell_1} = 1$, we should have $\sum_{\sigma \in S} f(\sigma) < 1/2$, where $S \subset S_n$ such that $\sigma^* \notin S$. Therefore, there is exactly one entry in each column of matrix $Q$ that is $> 1/2$, and the corresponding edge should be a part of $\sigma^*$. Thus, keeping only those edges $(i, j)$ such that $Q_{i,j} > 1/2$, we should the matching $\sigma^*$. It is clear from the construction that $\sigma^*$ indeed has the maximum weight of all the other matchings. The result now follows.

## 6  Conclusion

In summary, we considered the problem of inferring popular rankings from highly constrained information. Since raking data naturally arises in several diverse practical situations, an answer to this question has wide ranging implications.

Specifically, we considered the problem of inferring a sparse normalized function on the symmetric group using only the *first order* information about the function. In the election example this first order information corresponds to the fraction of people who have ranked candidate $i$ in the $j^{\text{th}}$ position. We provide a novel algorithm to precisely recover the permutations and the associated popularity under minimal, and essentially necessary, conditions. We provide justification to the necessity of our assumptions and consider a natural random model to quantify the sparsity that can be supported.

We also provide an algorithm, based on Fourier transform approximation, to determine the most popular ranking (mode of the function). The algorithm is essentially a max-weight matching with weights as the $q_{..}$ values. Under a natural majority assumption, the algorithm finds the correct mode.

The question considered is thematically related to harmonic analysis of functions over the symmetric group and also the currently popular topic of compressed sensing. The problem we consider can be restated as the reconstruction of a function using its *first order* Fourier representation, which has several applications particularly in the multi-object tracking problem. On the other hand, the parallels to the to the standard compressed sensing setup are limited because the available information is highly constrained. Thus, the existing approaches of compressed sensing cannot be applied to the problem.

**Next Steps.** We concentrated on the recovery of the distribution from its first order marginals. A possible next step would be to consider recovery under different forms of partial information. More specifically, practical applications motivate considering the recovery of distribution from pair-wise information: probability of candidate $i$ being ranked above candidate $j$. Another natural practical consideration would be to address the presence of noise in the available information. Understanding recovery of distributions with the above considerations are natural next steps.

## References

[1] C. Dwork, R. Kumar, M. Naor, and D. Sivakumar. Rank aggregation revisited. *In Proceedings of WWW10*, 2001.

[2] Yiling Chen, Lance Fortnow, Evdokia Nikolova, and David M. Pennock. Betting on permutations. In *EC '07: Proceedings of the 8th ACM conference on Electronic commerce*, pages 326–335, New York, NY, USA, 2007. ACM.

[3] J. Huang, C. Guestrin, and L. Guibas. Efficient Inference for Distributions on Permutations. *In Advances in Neural Information Processing Systems (NIPS)*, 2007.

[4] R. Kondor, A. Howard, and T. Jebara. Multi-object tracking with representations of the symmetric group. In *Proceedings of the Eleventh International Conference on Artificial Intelligence and Statistics*, 2007.

[5] P. Diaconis. Group Representations in Probability and Statistics. *IMS Lecture Notes-Monograph Series*, 11, 1988.

[6] E.J. Candes and T. Tao. Decoding by linear programming. *Information Theory, IEEE Transactions on*, 51(12):4203–4215, Dec. 2005.

[7] G. Birkhoff. Tres observaciones sobre el algebra lineal. *Univ. Nac. Tucuman Rev. Ser. A*, 5:147–151, 1946.

[8] J. Edmonds and R. Karp. Theoretical improvements in algorithmic efficiency for network flow problems. *Jour. of the ACM*, 19:248–264, 1972.

[9] M. Bayati, D. Shah, and M. Sharma. Max-product for maximum weight matching: convergence, correctness and lp duality. *IEEE Transactions on Information Theory*, March 2008.

